# Very loopy belief propagation for unwrapping phase images

**Brendan J. Frey[1], Ralf Koetter[2], Nemanja Petrovic[1,2]**

[1] Probabilistic and Statistical Inference Group, University of Toronto
http://www.psi.toronto.edu
[2] Electrical and Computer Engineering, University of Illinois at Urbana

## Abstract

Since the discovery that the best error-correcting decoding algorithm can be viewed as belief propagation in a cycle-bound graph, researchers have been trying to determine under what circumstances "loopy belief propagation" is effective for probabilistic inference. Despite several theoretical advances in our understanding of loopy belief propagation, to our knowledge, the only problem that has been solved using loopy belief propagation is error-correcting decoding on Gaussian channels. We propose a new representation for the two-dimensional phase unwrapping problem, and we show that loopy belief propagation produces results that are superior to existing techniques. This is an important result, since many imaging techniques, including magnetic resonance imaging and interferometric synthetic aperture radar, produce phase-wrapped images. Interestingly, the graph that we use has a very large number of very short cycles, supporting evidence that a large minimum cycle length is not needed for excellent results using belief propagation.

## 1 Introduction

Phase unwrapping is an easily stated, fundamental problem in image processing (Ghiglia and Pritt 1998). Each real-valued observation on a 1- or 2-dimensional grid is measured modulus a known wavelength, which we take to be 1 without loss of generality. Fig. 1b shows the wrapped, 1-dimensional waveform obtained from the original waveform shown in Fig. 1a. Every time the original waveform goes above 1 or below 0, it is wrapped to 0 or 1, respectively. The goal of phase unwrapping is to infer the original, unwrapped curve from the wrapped measurements, using using knowledge about which signals are more probable *a priori*.

In two dimensions, exact phase unwrapping is exponentially more difficult than 1-dimensional phase unwrapping and has been shown to be NP-hard in general (Chen and Zebker 2000). Fig. 1c shows the wrapped output of a magnetic resonance imaging device, courtesy of Z.-P. Liang. Notice the "fringe lines" – boundaries across which wrappings have occurred. Fig. 1d shows the wrapped terrain height measurements from an interferometric synthetic aperture radar, courtesy of Sandia National Laboratories, New Mexico.

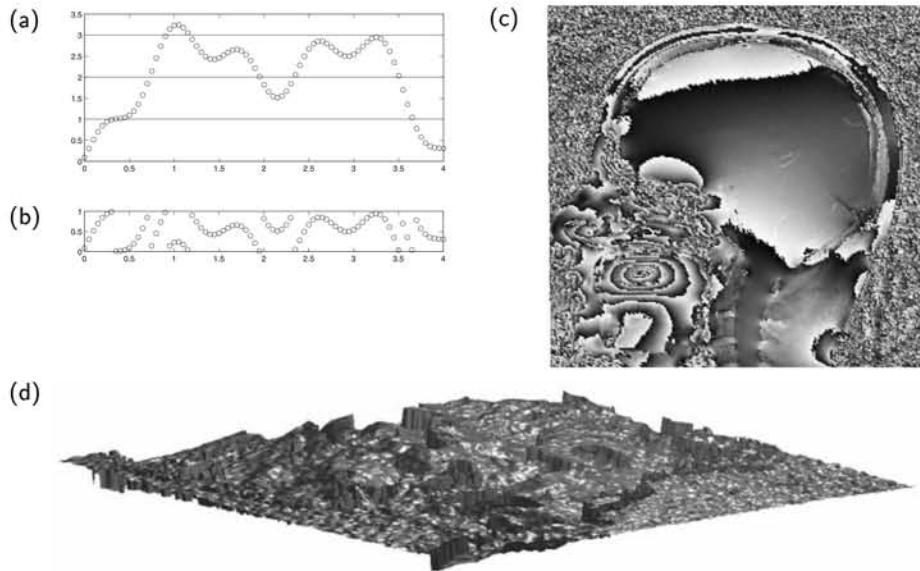

Figure 1: (a) A waveform measured on a 1-dimensional grid. (b) The phase-wrapped version of the waveform in (a), where the wavelength is 1. (c) A wrapped intensity map from a magnetic resonance imaging device, measured on a 2-dimensional grid (courtesy of Z.-P. Liang). (d) A wrapped topographic map measured on a 2-dimensional grid (courtesy of Sandia National Laboratories, New Mexico).

A sensible goal in phase unwrapping is to infer the gradient field of the original surface. The surface can then be reconstructed by integration. Equivalently, the goal is to infer the number of relative wrappings, or integer "shifts", between every pair of neighboring measurements. Positive shifts correspond to an increase in the number of wrappings in the direction of the $x$ or $y$ coordinate, whereas negative shifts correspond to a decrease in the number of wrappings in the direction of the $x$ or $y$ coordinate. After arbitrarily assigning an absolute number of wrappings to one point, the absolute number of wrappings at any other point can be determined by summing the shifts along a path connecting the two points. To account for direction, when taking a step against the direction of the coordinate, the shift should be subtracted.

When neighboring measurements are more likely closer together than farther apart *a priori*, 1-dimensional waveforms can be unwrapped optimally in time that is linear in the waveform length. For every pair of neighboring measurements, the shift that makes the unwrapped values as close together as possible is chosen. For example, the shift between 0.4 and 0.5 would be 0, whereas the shift between 0.9 and 0.0 would be $-1$.

For 2-dimensional surfaces and images, there are many possible 1-dimensional paths between any two points. These paths should be examined in combination, since the sum of the shifts along every such path should be equal. Viewing the shifts as state variables, the cut-set between any two points is exponential in the size of the grid, making exact inference for general priors NP-hard (Chen and Zebker 2000).

The two leading *fully-automated* techniques for phase unwrapping are the least squares method and the branch cut technique (Ghiglia and Pritt 1998). (Some other techniques perform better in some circumstances, but need additional information or require hand-tweaking.) The least squares method begins by making a greedy guess at the gradient between every pair of neighboring points. The resulting vector

field is not the gradient field of a surface, since in a valid gradient field, the sum of the gradients around every closed loop must be zero (that is, the *curl* must be 0). For example, the $2 \times 2$ loop of measurements 0.0, 0.3, 0.6, 0.9 will lead to gradients of 0.3, 0.3, 0.3, 0.1 around the loop, which do not sum to 0. The least squares method proceeds by projecting the vector field onto the linear subspace of gradient fields. The result is integrated to produce the surface. The branch cut technique also begins with greedy decisions for the gradients and then identifies untrustworthy regions of the image whose gradients should not be used during integration. As shown in our results section, both of these techniques are suboptimal.

Previously, we attempted to use a relaxed mean field technique to solve this problem (Achan, Frey and Koetter 2001). Here, we take a new approach that works better and is motivated by the impressive results of belief propagation in cycle-bound graphs for error-correcting decoding (Wiberg, Loeliger and Koetter 1995; MacKay and Neal 1995; Frey and Kschischang 1996; Kschischang and Frey 1998; McEliece, MacKay and Cheng 1998). In contrast to other work (Ghiglia and Pritt 1998; Chen and Zebker 2000; Koetter et al. 2001), we introduce a new framework for quantitative evaluation, which impressively places belief propagation much closer to the theoretical limit than other leading methods.

It is well-known that belief propagation (a.k.a. the sum-product algorithm, probability propagation) is exact in graphs that are trees (Pearl 1988), but it has been discovered only recently that it can produce excellent results in graphs with many cycles. Impressive results have been obtained using loopy belief propagation for super-resolution (Freeman and Pasztor 1999) and for infering layered representations of scenes (Frey 2000). However, despite several theoretical advances in our understanding of loopy belief propagation (c.f. (Weiss and Freeman 2001)) and proposals for modifications to the algorithm (c.f. (Yedidia, Freeman and Weiss 2001)), to our knowledge, the only problem that has been solved by loopy belief propagation is error-correcting decoding on Gaussian channels.

*We conjecture that although phase unwrapping is generally NP-hard, there exists a near-optimal phase unwrapping algorithm for Gaussian process priors. Further, we believe that algorithm to be loopy belief propagation.*

## 2   Loopy Belief Propagation for Phase Unwrapping

As described above, the goal is to infer the number of relative wrappings, or integer "shifts", between every pair of neighboring measurements. Denote the $x$-direction shift at $(x, y)$ by $a(x, y)$ and the $y$-direction shift at $(x, y)$ by $b(x, y)$, as shown in Fig. 2a. If the sum of the shifts around every short loop of 4 shifts (*e.g.*, $a(x, y) + b(x + 1, y) - a(x, y + 1) - b(x, y)$ in Fig. 2a) is zero, then perturbing a path will not change the sum of the shifts along the path. So, a valid set of shifts $\mathcal{S} = \{a(x, y), b(x, y) : x = 1, \ldots, N - 1; y = 1, \ldots, M - 1\}$ in an $N \times M$ image must satisfy the constraint

$$a(x, y) + b(x + 1, y) - a(x, y + 1) - b(x, y) = 0, \qquad (1)$$

for $x = 1, \ldots, N-1, y = 1, \ldots, M-1$. Since $a(x, y) + b(x+1, y) - a(x, y+1) - b(x, y)$ is a measure of curl at $(x, y)$, we refer to (1) as a "zero-curl constraint", reflecting the fact that the curl of a gradient field is 0. In this way, phase unwrapping is formulated as the problem of inferring the most probable set of shifts subject to satisfying all zero-curl constraints.

We assume that given the set of shifts, the unwrapped surface is described by a low-order Gaussian process. The joint distribution over the shifts $\mathcal{S} = \{a(x, y), b(x, y) : x = 1, \ldots, N - 1; y = 1, \ldots, M - 1\}$ and the wrapped measurements $\Phi = \{\phi(x, y) :$

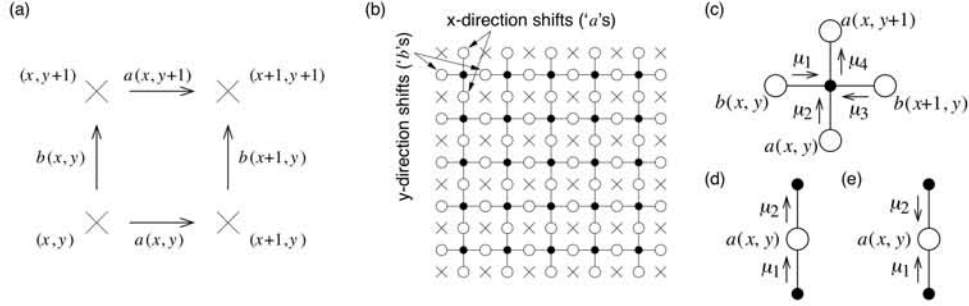

Figure 2: (a) Positive $x$-direction shifts (arrows labeled $a$) and positive $y$-direction shifts (arrows labeled $b$) between neighboring measurements in a $2 \times 2$ patch of points (marked by $\times$'s). (b) A graphical model that describes the zero-curl constraints (black discs) between neighboring shift variables (white discs). 3-element probability vectors ($\boldsymbol{\mu}$'s) on the relative shifts between neighboring variables ($-1$, $0$, or $+1$) are propagated across the network: (c) Constraint-to-shift vectors are computed from incoming shift-to-constraint vectors; (d) Shift-to-constraint vectors are computed from incoming constraint-to-shift vectors; (d) Estimates of the marginal probabilities of the shifts given the data are computed by combining incoming constraint-to-shift vectors.

$0 \le \phi(x,y) < 1, x = 1, \ldots, N; y = 1, \ldots, M\}$ can be expressed in the form

$$P(\mathcal{S}, \Phi) \propto \prod_{x=1}^{N-1} \prod_{y=1}^{M-1} \delta(a(x,y) + b(x+1,y) - a(x,y+1) - b(x,y))$$

$$\cdot \prod_{x=1}^{N-1} \prod_{y=1}^{M} e^{-(\phi(x+1,y)-\phi(x,y)-a(x,y))^2/2\sigma^2} \prod_{x=1}^{N} \prod_{y=1}^{M-1} e^{-(\phi(x,y+1)-\phi(x,y)-b(x,y))^2/2\sigma^2}. \quad (2)$$

The zero-curl constraints are enforced by $\delta(\cdot)$, which evaluates to 1 if its argument is 0 and evaluates to 0 otherwise. We assume the slope of the surface is limited so that the unknown shifts take on the values $-1$, $0$ and $1$. $\sigma^2$ is the variance between two neighboring measurements in the unwrapped image, but we find that in practice it can be estimated directly from the wrapped image.

Phase unwrapping consists of making inferences about the $a$'s and $b$'s in the above probability model. For example, the marginal probability that the $x$-direction shift at $(x,y)$ is $k$ given an observed wrapped image $\Phi$, is

$$P(a(x,y) = k|\Phi) \propto \sum_{\mathcal{S}:a(x,y)=k} P(\mathcal{S}, \Phi). \quad (3)$$

For an $N \times M$ grid, the above sum has roughly $3^{2NM}$ terms and so exact inference is intractable.

The factorization of the joint distribution in (2) can be described by a graphical model, as shown in Fig. 2b. In this graph, each white disc sits on the border between two measurements (marked by $\times$'s), and corresponds to either an $x$-direction shift ($a$'s) or a $y$-direction shift ($b$'s). Each black disc corresponds to a zero-curl constraint ($\delta(\cdot)$ in (2)), and is connected to the 4 shifts that it constrains to sum to 0.

Probability propagation computes messages (3-vectors denoted by $\boldsymbol{\mu}$) which are passed in both directions on every edge in the network. The elements of each 3-vector correspond to the allowed values of the neighboring shift, $-1$, $0$ and $1$. Each of these 3-vectors can be thought of as a probability distribution over the 3 possible values that the shift can take on.

Each element in a constraint-to-shift message summarizes the evidence from the other 3 shifts involved in the constraint, and is computed by averaging the allowed configurations of evidence from the other 3 shifts in the constraint. For example, if $\mu_1$, $\mu_2$ and $\mu_3$ are 3-vectors entering a constraint as shown in Fig. 2c, the outgoing 3-vector, $\mu_4$, is computed using

$$\mu_{4i} = \sum_{j=-1}^{1} \sum_{k=-1}^{1} \sum_{l=-1}^{1} \delta(k + l - i - j)\mu_{1j}\mu_{2k}\mu_{3l}, \qquad (4)$$

and then normalized for numerical stability. The other 3 messages produced at the constraint are computed in a similar fashion.

Shift-to-constraint messages are computed by weighting incoming constraint-to-shift messages with the likelihood for the shift. For example, if $\mu_1$ is a 3-vector entering an $x$-direction shift as shown in Fig. 2d, the outgoing 3-vector, $\mu_2$ is computed using

$$\mu_{2i} = \mu_{1i} \exp[-(\phi(x + 1, y) - \phi(x, y) - i)^2 / 2\sigma^2], \qquad (5)$$

and then normalized. Messages produced by $y$-direction shifts are computed in a similar fashion.

At any step in the message-passing process, the messages on the edges connected to a shift variable can be combined to produce an approximation to the marginal probability for that shift, given the observations. For example, if $\mu_1$ and $\mu_2$ are the 3-vectors entering an $x$-direction shift as shown in Fig. 2e, the approximation is

$$\hat{P}(a(x,y) = i|\Phi) = (\mu_{1i}\mu_{2i}) / (\sum_{j} \mu_{1j}\mu_{2j}). \qquad (6)$$

Given a wrapped image, the variance $\sigma^2$ is estimated directly from the wrapped image, the probability vectors are initialized to uniform distributions, and then probability vectors are propagated across the graph in an iterative fashion. Different message-passing schedules are possible, ranging from fully parallel, to a "forward-backward-up-down"-type schedule, in which messages are passed across the network to the right, then to the left, then up and then down. For an $N \times M$ grid, each iteration takes $\mathcal{O}(NM)$ scalar computations.

After probability propagation converges (or, after a fixed number of iterations), estimates of the marginal probabilities of the shifts given the data are computed, and the most probable value of each shift variable is selected. The resulting configuration of the shifts can then be integrated to obtain the unwrapped surface. If some zero-curl constraints remain violated, a robust integration technique, such as least squares integration (Ghiglia and Pritt 1998), should be used.

## 3 Experimental results

Generally, belief propagation in cycle-bound graphs is not guaranteed to converge. Even if it does converge, the approximate marginals may not be close to the true marginals. So, the algorithm must be verified by experiments.

On surfaces drawn from Gaussian process priors, we find that the belief propagation algorithm produces significantly lower reconstruction errors than the least squares method and the branch cut technique.

Here, we focus on the performances of the algorithms for real data recorded from a synthetic aperture radar device (Fig. 1d). Since our algorithm assumes the surface is Gaussian given the shifts, a valid concern is that it will not perform well when the Gaussian process prior is incorrect.

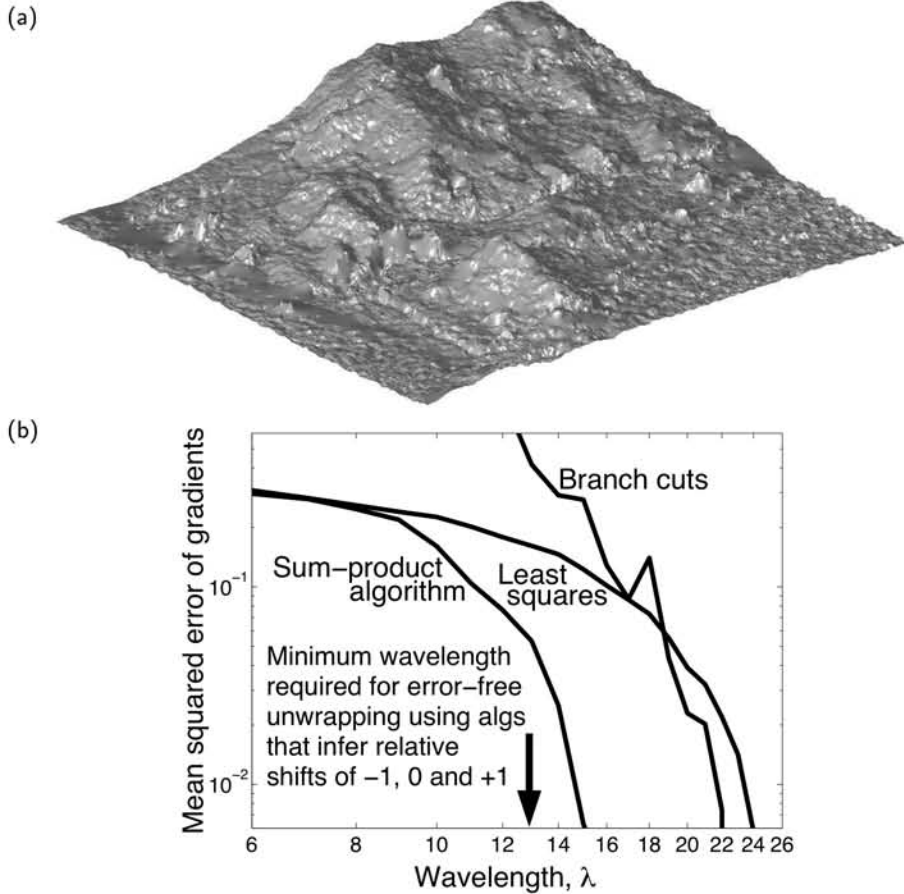

(a)

(b)

Figure 3: (a) After 10 iterations of belief propagation using the phase-wrapped surface from Fig. 1d, hard decisions were made for the shift variables and the resulting shifts were integrated to produce this unwrapped surface. (b) Reconstruction error versus wrapping wavelength for our technique, the least squares method and the branch cuts technique.

Fig. 3a shows the surface that is obtained by setting $\sigma^2$ to the mean squared difference between neighboring wrapped values, applying 10 iterations of belief propagation, making hard decisions for the integer shifts, and integrating the resulting gradients. Since this is real data, we do not know the "ground truth". However, compared to the least squares method, our algorithm preserves more detail. The branch cut technique is not able to unwrap the entire surface.

To obtain quantitative results on reconstruction error, we use the surface produced by the least squares method as the "ground truth". To determine the effect of wrapping wavelength on algorithm performance, we rewrap this surface using different wavelengths. For each wavelength, we compute the reconstruction error for belief propagation, least squares and branch cuts. Note that by using least squares to obtain the ground truth, we may be biasing our results *in favor* of least squares.

Fig. 3b shows the logarithm of the mean squared error in the gradient field of the reconstructed surface as a function of the wrapping wavelength, $\lambda$, on a log-scale. (The plot for the mean squared error in the surface heights looks similar.) As $\lambda \to 0$, unwrapping becomes impossible and as $\lambda \to \infty$, unwrapping becomes trivial (since no wrappings occur), so algorithms have waterfall-shaped curves.

The belief propagation algorithm clearly obtains significantly lower reconstruction errors. Viewed another way, belief propagation can tolerate much lower wrapping wavelengths for a given reconstruction error. Also, it turns out that for this surface, it is impossible for an algorithm that infers relative shifts of $-1$, $0$ and $1$ to obtain a reconstruction error of $0$, unless $\lambda \geq 12.97$. Belief propagation obtains a zero-error wavelength that is significantly closer to this limit than the least squares method and the branch cuts technique.

## 4 Conclusions

Phase unwrapping is a fundamental problem in image processing and although it has been shown to be NP-hard for general priors (Chen and Zebker 2000), we conjecture there exists a near-optimal phase unwrapping algorithm for Gaussian process priors. Further, we believe that algorithm to be loopy belief propagation. Our experimental results show that loopy belief propagation obtains significantly lower reconstruction errors compared to the least squares method and the branch cuts technique (Ghiglia and Pritt 1998), and performs close to the theoretical limit for techniques that infer relative wrappings of $-1$, $0$ and $+1$. The belief propagation algorithm runs in about the same time as the other techniques.

## References

Achan, K., Frey, B. J., and Koetter, R. 2001. A factorized variational technique for phase unwrapping in Markov random fields. In *Uncertainty in Artificial Intelligence 2001*. Seattle, Washington.

Chen, C. W. and Zebker, H. A. 2000. Network approaches to two-dimensional phase unwrapping: intractability and two new algorithms. *Journal of the Optical Society of America A*, 17(3):401–414.

Freeman, W. and Pasztor, E. 1999. Learning low-level vision. In *Proceedings of the International Conference on Computer Vision*, pages 1182–1189.

Frey, B. J. 2000. Filling in scenes by propagating probabilities through layers and into appearance models. In *Proceedings of the IEEE Conference on Computer Vision and Pattern Recognition*.

Frey, B. J. and Kschischang, F. R. 1996. Probability propagation and iterative decoding. In *Proceedings of the 34th Allerton Conference on Communication, Control and Computing 1996*.

Ghiglia, D. C. and Pritt, M. D. 1998. *Two-Dimensional Phase Unwrapping. Theory, Algorithms and Software*. John Wiley & Sons.

Koetter, R., Frey, B. J., Petrovic, N., and Munson, Jr., D. C. 2001. Unwrapping phase images by propagating probabilities across graphs. In *Proceedings of the International Conference on Acoustics, Speech and Signal Processing*. IEEE Press.

Kschischang, F. R. and Frey, B. J. 1998. Iterative decoding of compound codes by probability propagation in graphical models. *IEEE Journal on Selected Areas in Communications*, 16(2):219–230.

MacKay, D. J. C. and Neal, R. M. 1995. Good codes based on very sparse matrices. In Boyd, C., editor, *Cryptography and Coding. 5th IMA Conference*, number 1025 in Lecture Notes in Computer Science, pages 100–111. Springer, Berlin Germany.

McEliece, R. J., MacKay, D. J. C., and Cheng, J. F. 1998. Turbo-decoding as an instance of Pearl's 'belief propagation' algorithm. *IEEE Journal on Selected Areas in Communications*, 16.

Pearl, J. 1988. *Probabilistic Reasoning in Intelligent Systems*. Morgan Kaufmann, San Mateo CA.

Weiss, Y. and Freeman, W. 2001. On the optimaility of solutions of the max-product belief propagation algorithm in artbitrary graphs. *IEEE Transactions on Information Theory, Special Issue on Codes on Graphs and Iterative Algorithms*, 47(2):736–744.

Wiberg, N., Loeliger, H.-A., and Koetter, R. 1995. Codes and iterative decoding on general graphs. *European Transactions on Telecommunications*, 6:513–525.

Yedidia, J., Freeman, W. T., and Weiss, Y. 2001. Generalized belief propagation. In *Advances in Neural Information Processing Systems 13*. MIT Press, Cambridge MA.
